# Seeing through water

**Alexei A. Efros**[*]
School of Computer Science
Carnegie Mellon University
Pittsburgh, PA 15213, U.S.A.
efros@cs.cmu.edu

**Volkan Isler, Jianbo Shi and Mirkó Visontai**
Dept. of Computer and Information Science
University of Pennsylvania
Philadelphia, PA 19104
{isleri,jshi,mirko}@cis.upenn.edu

## Abstract

We consider the problem of recovering an underwater image distorted by surface waves. A large amount of video data of the distorted image is acquired. The problem is posed in terms of finding an undistorted image patch at each spatial location. This challenging reconstruction task can be formulated as a manifold learning problem, such that the center of the manifold is the image of the undistorted patch. To compute the center, we present a new technique to estimate global distances on the manifold. Our technique achieves robustness through *convex flow* computations and solves the "leakage" problem inherent in recent manifold embedding techniques.

## 1 Introduction

Consider the following problem. A pool of water is observed by a stationary video camera mounted above the pool and looking straight down. There are waves on the surface of the water and all the camera sees is a series of distorted images of the bottom of the pool, e.g. Figure 1. The aim is to use these images to recover the undistorted image of the pool floor – as if the water was perfectly still. Besides obvious applications in ocean optics and underwater imaging [1], variants of this problem also arise in several other fields, including astronomy (overcoming atmospheric distortions) and structure-from-motion (learning the appearance of a deforming object). Most approaches to solve this problem try to model the distortions explicitly. In order to do this, it is critical not only to have a good parametric model of the distortion process, but also to be able to reliably extract features from the data to fit the parameters. As such, this approach is only feasible in well understood, highly controlled domains. On the opposite side of the spectrum is a very simple method used in underwater imaging: simply, average the data temporally. Although this method performs surprisingly well in many situations, it fails when the structure of the target image is too fine with respect to the amplitude of the wave (Figure 2).

In this paper we propose to look at this difficult problem from a more statistical angle. We will exploit a very simple observation: if we watch a particular spot on the image plane, most of the time the picture projected there will be distorted. But once in a while, when the water just happens to be locally flat at that point, we will be looking straight down and seeing exactly the right spot on the ground. If we can recognize when this happens

---

[*]Authors in alphabetical order.

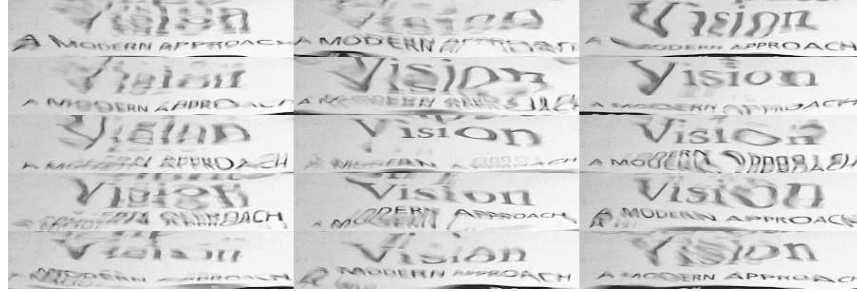

Figure 1: Fifteen consecutive frames from the video. The experimental setup involved: a transparent bucket of water, the cover of a vision textbook "Computer Vision/A Modern Approach".

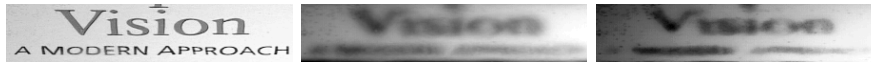

Figure 2: Ground truth image and reconstruction results using mean and median

and snap the right picture at each spatial location, then recovering the desired ground truth image would be simply a matter of stitching these correct observations together. In other words, the question that we will be exploring in this paper is not *where* to look, but *when*!

## 2 Problem setup

Let us first examine the physical setup of our problem. There is a "ground truth" image $G$ on the bottom of the pool. Overhead, a stationary camera pointing downwards is recording a video stream $V$. In the absence of any distortion $V(x, y, t) = G(x, y)$ at any time $t$. However, the water surface refracts in accordance with Snell's Law. Let us consider what the camera is seeing at a particular point $x$ on the CCD array, as shown in Figure 3(c) (assume 1D for simplicity). If the normal to the water surface directly underneath $x$ is pointing straight up, there is no refraction and $V(x) = G(x)$. However, if the normal is tilted by angle $\theta_1$, light will bend by the amount $\theta_2 = \theta_1 - \sin^{-1}\left(\frac{1}{1.33}\sin\theta_1\right)$, so the camera point $V(x)$ will see the light projected from $G(x + dx)$ on the ground plane. It is easy to see that the relationship between the tilt of the normal to the surface $\theta_1$ and the displacement $dx$ is approximately linear ($dx \approx 0.25\theta_1 h$ using small angle approximation, where $h$ is the height of the water). This means that, in 2D, what the camera will be seeing over time at point $V(x, y, t)$ are points on the ground plane sampled from a disk centered at $G(x, y)$ and with radius related to the height of the water and the overall roughness of the water surface. A similar relationship holds in the inverse direction as well: a point $G(x, y)$ will be imaged on a disk centered around $V(x, y)$.

What about the distribution of these sample points? According to Cox-Munk Law [2], the surface normals of rough water are distributed approximately as a Gaussian centered around the vertical, assuming a large surface area and stationary waves. Our own experiments, conducted by hand-tracking (Figure 3b), confirm that the distribution, though not exactly Gaussian, is definitely unimodal and smooth.

Up to now, we only concerned ourselves with infinitesimally small points on the image or the ground plane. However, in practice, we must have something that we can compute with. Therefore, we will make an assumption that the surface of the water can be locally approximated by a planar patch. This means that everything that was true for points is now true for local image patches (up to a small affine distortion).

# 3 Tracking via embedding

From the description outlined above, one possible solution emerges. If the distribution of a particular ground point on the image plane is unimodal, then one could track feature points in the video sequence over time. Computing their mean positions over the entire video will give an estimate of their true positions on the ground plane. Unfortunately, tracking over long periods of time is difficult even under favorable conditions, whereas our data is so fast (undersampled) and noisy that reliable tracking is out of the question (Figure 3(c)).

However, since we have a lot of data, we can substitute smoothness in time with *smoothness in similarity* – for a given patch we are more likely to find a patch similar to it *somewhere* in time, and will have a better chance to track the transition between them. An alternative to tracking the patches directly (which amounts to holding the ground patch $G(x, y)$ fixed and centering the image patches $V(x+dx_t, y+dy_t)$ on top of it in each frame), is to fix the image patch $V(x, y)$ in space and observe the patches from $G(x + dx_t, y + dy_t)$ appearing in this window. We know that this set of patches comes from a disk on the ground plane centered around patch $G(x, y)$ – our goal. If the disk was small enough compared to the size of the patch, we could just cluster the patches together, e.g. by using translational EM [3]. Unfortunately, the disk can be rather large, containing patches with no overlap at all, thus making only the local similarity comparisons possible. However, notice that our set of patches lies on a low-dimensional manifold; in fact we know precisely which manifold – it's the disk on the ground plane centered at $G(x, y)$! So, if we could use the local patch similarities to find an embedding of the patches in $V(x, y, t)$ on this manifold, the center of the embedding will hold our desired patch $G(x, y)$.

The problem of embedding the patches based on local similarity is related to the recent work in manifold learning [4, 5]. Basic ingredients of the embedding algorithms are: defining a distance measure between points, and finding an energy function that optimally places them in the embedding space. The distance can be defined as all-pairs distance matrix, or as distance from a particular reference node. In both cases, we want the distance function to satisfy some constraints to model the underlying physical problem.

The local similarity measure for our problem turned out to be particularly unreliable, so none of the previous manifold learning techniques were adequate for our purposes. In the following section we will describe our own, robust method for computing a global distance function and finding the right embedding and eventually the center of it.

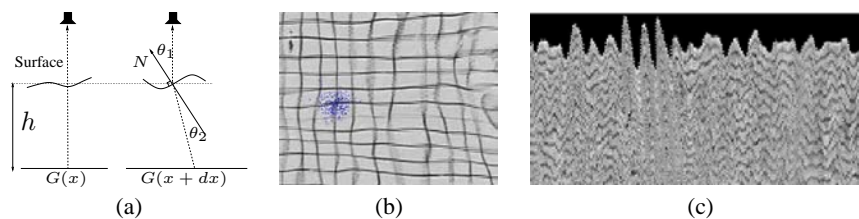

(a)  (b)  (c)

Figure 3: (a) Snell's Law (b)-(c) Tracking points of the bottom of the pool: (b) the tracked position forms a distribution close to a Gaussian, (c): a vertical line of the image shown at different time instances (horizontal axis). The discontinuity caused by rapid changes makes the tracking infeasible.

# 4 What is the right distance function?

Let $\mathcal{I} = \{I_1, \ldots, I_n\}$ be the set of patches, where $I_t = V(x, y, t)$ and $x = [x_{min}, x_{max}], y = [y_{min}, y_{max}]$ are the patch pixel coordinates. Our goal is to find a *center* patch to represent the set $\mathcal{I}$. To achieve this goal, we need a distance function

$d : \mathcal{I} \times \mathcal{I} \rightarrow \mathbb{R}$ such that $d(I_i, I_j) < d(I_i, I_k)$ implies that $I_j$ is more similar to $I_i$ than $I_k$. Once we have such a measure, the center can be found by computing:

$$I^* = \arg\min_{I_i \in \mathcal{I}} \sum_{I_j \in \mathcal{I}} d(I_i, I_j) \tag{1}$$

Unfortunately, the measurable distance functions, such as Normalized Cross Correlation ($NCC$) are only local. A common approach is to design a global distance function using the measurable local distances and *transitivity* [6, 4]. This is equivalent to designing a global distance function of the form:

$$d(I_i, I_j) = \begin{cases} d_{local}(I_i, I_j), & \text{if } d_{local}(I_i, I_j) \leq \tau \\ d_{transitive}(I_i, I_j), & \text{otherwise.} \end{cases} \tag{2}$$

where $d_{local}$ is a local distance function, $\tau$ is a user-specified threshold and $d_{transitive}$ is a global, transitive distance function which utilizes $d_{local}$. The underlying assumption here is that the members of $\mathcal{I}$ lie on a constraint space (or manifold) $\mathcal{S}$. Hence, a local similarity function such as $NCC$ can be used to measure local distances on the manifold. An important research question in machine learning is to extend the local measurements into global ones, i.e. to design $d_{transitive}$ above.

One method for designing such a transitive distance function is to build a graph $G = (V, E)$ whose vertices correspond to the members of $\mathcal{I}$. The local distance measure is used to place edges which connect only very similar members of $\mathcal{I}$. Afterwards, the length of pairwise shortest paths are used to estimate the true distances on the manifold $\mathcal{S}$. For example, this method forms the basis of the well-known Isomap method [4].

Unfortunately, estimating the distance $d_{transitive}(\cdot, \cdot)$ using shortest path computations is not robust to errors in the local distances – which are very common. Consider a patch that contains the letter A and another one that contains the letter B. Since they are different letters, we expect that these patches would be quite distant on the manifold $\mathcal{S}$. However, among the A patches there will inevitably be a very blurry A that would look quite similar to a very blurry B producing an erroneous local distance measurement. When the transitive global distances are computed using shortest paths, a single erroneous edge will single-handedly cause *all* the A patches to be much closer to *all* the B patches, short-circuiting the graph and completely distorting all the distances.

Such errors lead to the *leakage problem* in estimating the global distances of patches. This problem is illustrated in Figure 4. In this example, our underlying manifold $\mathcal{S}$ is a triangle. Suppose our local distance function erroneously estimates an edge between the corners of the triangle as shown in the figure. After the erroneous edge is inserted, the shortest paths from the top of the triangle *leak* through this edge. Therefore, the shortest path distances will fail to reflect the true distance on the manifold.

## 5    Solving the leakage problem

Recall that our goal is to find the center of our data set as defined in Equation 1. Note that, in order to compute the center we do not need all pairwise distances. All we need is the quantity $d_{\mathcal{I}}(I_i) = \sum_{I_j \in \mathcal{I}} d(I_i, I_j)$ for all $I_i$.

The leakage problem occurs when we compute the values $d_{\mathcal{I}}(I_i)$ using the shortest path metric. In this case, even a single erroneous edge may reduce the shortest paths from many different patches to $I_i$ – changing the value of $d_{\mathcal{I}}(I_i)$ drastically. Intuitively, in order to prevent the leakage problem we must prevent edges from getting involved in many shortest path computations to the same node (i.e. leaking edges). We can formalize this notion by casting the computation as a network flow problem.

Let $G = (V, E)$ be our graph representation such that for each patch $I_i \in \mathcal{I}$, there is a vertex $v_i \in V$. The edge set $E$ is built as follows: there is an edge $(v_i, v_j)$ if $d_{local}(I_i, I_j)$ is less than a threshold. The *weight* of the edge $(v_i, v_j)$ is equal to $d_{local}(I_i, I_j)$.

To compute the value $d_{\mathcal{I}}(I_i)$, we build a flow network whose vertex set is also $V$. All vertices in $V - \{v_i\}$ are sources, pushing unit flow into the network. The vertex $v_i$ is a sink with infinite capacity. The arcs of the flow network are chosen using the edge set $E$. For each edge $(v_j, v_k) \in E$ we add the arcs $v_j \rightarrow v_k$ and $v_k \rightarrow v_j$. Both arcs have infinite capacity and the cost of pushing one unit of flow on either arc is equal to the weight of $(v_j, v_k)$, as shown in Figure 4 left (top and bottom). It can easily be seen that the minimum cost flow in this network is equal to $d_{\mathcal{I}}(I_i)$. Let us call this network which is used to compute $d_{\mathcal{I}}(I_i)$ as $NW(I_i)$.

The crucial factor in designing such a flow network is choosing the right *cost* and *capacity*. Computing the minimum cost flow on $NW(I_i)$ not only gives us $d_{\mathcal{I}}(I_i)$ but also allows us to compute how many times an edge is involved in the distance computation: the amount of flow through an edge is exactly the number of times that edge is used for the shortest path computations. This is illustrated in Figure 4 (box A) where $d_1$ units of cost is charged for each unit of flow through the edge $(u, w)$. Therefore, if we prevent too much flow going through an edge, we can prevent the leakage problem.

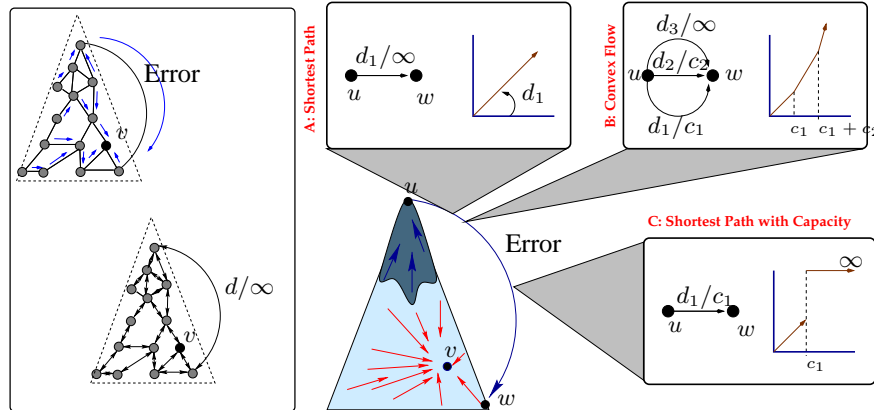

Figure 4: The leakage problem. **Left:** Equivalence of shortest path leakage and uncapacitated flow leakage problem. **Bottom-middle:** After the erroneous edge is inserted, the shortest paths from the top of the triangle to vertex $v$ go through this edge. **Boxes A-C:** Alternatives for charging a unit of flow between nodes $u$ and $w$. The horizontal axis of the plots is the amount of flow and the vertical axis is the cost. **Box A:** Linear flow. The cost of a unit of flow is $d_1$ **Box B:** Convex flow. Multiple edges are introduced between two nodes, with fixed capacity, and convexly increasing costs. The cost of a unit of flow increases from $d_1$ to $d_2$ and then to $d_3$ as the amount of flow from $u$ to $w$ increases. **Box C:** Linear flow with capacity. The cost is $d_1$ until a capacity of $c_1$ is achieved and becomes infinite afterwards.

One might think that the leakage problem can simply be avoided by imposing capacity constraints on the arcs of the flow network (Figure 4, box C). Unfortunately, this is not very easy. Observe that in the minimum cost flow solution of the network $NW(I_i)$, the amount of flow on the arcs will increase as the arcs get closer to $I_i$. Therefore, when we are setting up the network $NW(I_i)$, we must adaptively increase the capacities of arcs "closer" to the sink $v_i$ – otherwise, there will be no feasible solution. As the structure of the graph $G$ gets complicated, specifying this notion of closeness becomes a subtle issue. Further, the structure of the underlying space $\mathcal{S}$ could be such that some arcs in $G$ must indeed

carry a lot of flow. Therefore imposing capacities on the arcs requires understanding the underlying structure of the graph $G$ as well as the space $\mathcal{S}$ – which is in fact the problem we are trying to solve!

Our proposed solution to the leakage problem uses the notion of a *convex flow*. We do not impose a capacity on the arcs. Instead, we impose a convex cost function on the arcs such that the cost of pushing unit flow on arc $a$ increases as the total amount of flow through $a$ increases. See Figure 4, box B.

This can be achieved by transforming the network $NW(I_i)$ to a new network $NW'(I_i)$. The transformation is achieved by applying the following operation on each arc in $NW(I_i)$: Let $a$ be an arc from $u$ to $w$ in $NW(I_i)$. In $NW'(I_i)$, we replace $a$ by $k$ arcs $a_1, \ldots, a_k$. The costs of these arcs are chosen to be uniformly increasing so that $cost(a_1) < cost(a_2) < \ldots < cost(a_k)$. The capacity of arc $a_k$ is infinite. The weights and capacities of the other arcs are chosen to reflect the steepness of the desired convexity (Figure 4, box B). The network shown in the figure yields the following function for the cost of pushing $x$ units of flow through the arc:

$$cost(x) = \begin{cases} d_1 x, & \text{if } 0 \le x \le c_1 \\ d_1 c_1 + d_2(x - c_1), & \text{if } c_1 \le x \le c_2 \\ d_1 c_1 + d_2(c_2 - c_1) + d_3(x - c_1 - c_2), & \text{if } c_2 \le x \end{cases} \qquad (3)$$

The advantage of this convex flow computation is twofold. It does not require putting thresholds on the arcs a-priori. It is always feasible to have as much flow on a single arc as required. However, the minimum cost flow will avoid the leakage problem because it will be costly to use an erroneous edge to carry the flow from many different patches.

## 5.1 Fixing the leakage in Isomap

As noted earlier, the Isomap method [4] uses the shortest path measurements to estimate a distance matrix $M$. Afterwards, $M$ is used to find an embedding of the manifold $\mathcal{S}$ via MDS.

As expected, this method also suffers from the leakage problem as demonstrated in Figure 5. The top-left image in Figure 5 shows our ground truth. In the middle row, we present an embedding of these graphs computed using Isomap which uses the shortest path length as the global distance measure. As illustrated in these figures, even though isomap does a good job in embedding the ground truth when there are no errors, the embedding (or manifold) collapses after we insert the erroneous edges. In contrast, when we use the convex-flow based technique to estimate the distances, we recover the true embedding – even in the presence of erroneous edges (Figure 5 bottom row).

## 6 Results

In our experiments we used $800$ image frames to reconstruct the ground truth image. We fixed $30 \times 30$ size patches in each frame at the same location (see top of Figure 7 for two sets of examples), and for every location we found the center. The middle row of Figure 7 shows embeddings of the patches computed using the distance derived from the convex flow. The transition path and the morphing from selected patches (A,B,C) to the center patch (F) is shown at the bottom.

The embedding plot on the left is considered an easier case, with a Gaussian-like embedding (the graph is denser close to the center) and smooth transitions between the patches in a transition path. The plot to the right shows a more difficult example, when the embedding has no longer a Gaussian shape, but rather a triangular one. Also note that the transitions can have jumps connecting non-similar patches which are distant in the embedding space. The two extremes of the triangle represent the blurry patches, which are so numerous and

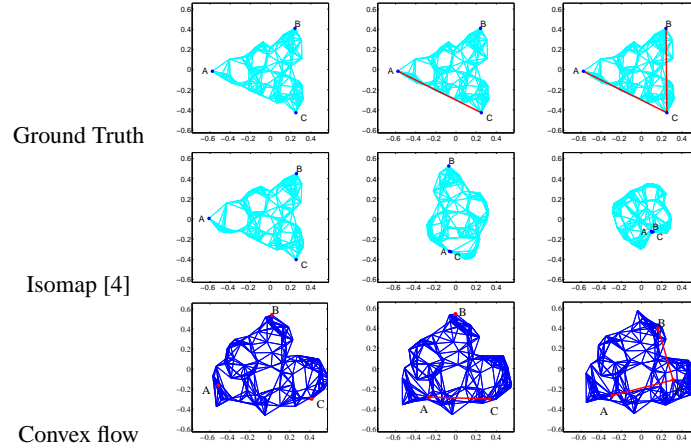

Figure 5: **Top row:** Ground truth. After sampling points from a triangular disk, a kNN graph is constructed to provide a local measure for the embedding (left). Additional erroneous edges $AC$ and $CB$ are added to perturb the local measure (middle, right). **Middle row:** Isomap embedding. Isomap recovers the manifold for the error-free cases (left). However, all-pairs shortest path can "leak" through $AC$ and $CB$, resulting a significant change in the embedding. **Bottom row:** Convex flow embedding. Convex flow penalized too many paths going through the same edge – correcting the leakage problem. The resulting embedding is more resistant to perturbations in the kNN graph.

very similar to each other, so that they are no longer treated as noise or outliers. This results in 'folding in' the embedding and thus, moving estimated the center towards the blurry patches. To solve this problem, we introduced additional two centers, which ideally would represent the blurry patches, allowing the third center to move to the ground truth.

Once we have found the centers for all patches we stitched them together to form the complete reconstructed image. In case of three centers, we use overlapping patches and dynamic programming to determine the best stitching. Figure 6 shows the reconstruction

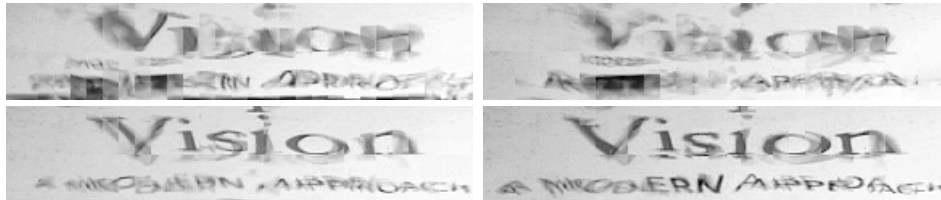

Figure 6: Comparison of reconstruction results of different methods using the first 800 frames, **top**: patches stitched together which are closest to mean (**left**) and median (**right**), **bottom**: our results using a single (**left**) and three (**right**) centers

result of our algorithm compared to simple methods of taking the mean/median of the patches and finding the closest patch to them. The bottom row shows our result for a single and for three center patches. The better performance of the latter suggests that the two new centers relieve the correct center from the blurry patches.

For a graph with $n$ vertices and $m$ edges, the minimum cost flow computation takes $O(m \log n(m + n \log n))$ time, therefore finding the center $I^*$ of one set of patches can be done in $O(mn \log n(m + n \log n))$ time. Our flow computation is based on the min-cost max-flow implementation by Goldberg [7]. The convex function used in our experiments was as described in Equation 3 with parameters $d_1 = 1$, $c_1 = 1$, $d_2 = 5$, $c_2 = 9$, $d_3 = 50$.

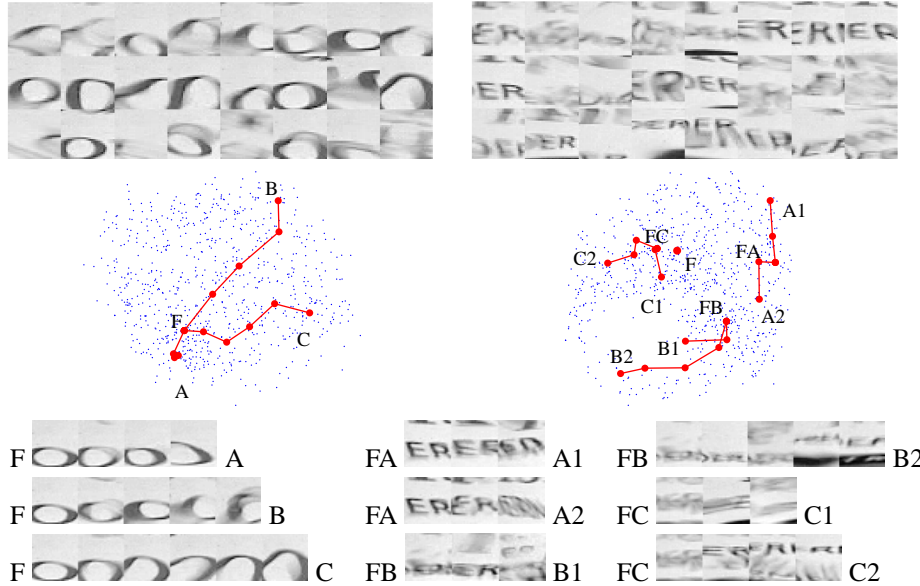

Figure 7: **Top row:** sample patches (two different locations) from 800 frames, **Middle row:** Convex flow embedding, showing the transition paths. **Bottom row:** corresponding patches (A, B, C, A1, A2, B1, B2, C1, C2) and the morphing of them to the centers F F, FA, FB, FC respectively

## 7   Conclusion

In this paper, we studied the problem of recovering an underwater image from a video sequence. Because of the surface waves, the sequence consists of distorted versions of the image to be recovered. The novelty of our work is in the formulation of the reconstruction problem as a manifold embedding problem. Our contribution also includes a new technique, based on *convex flows*, to recover global distances on the manifold in a robust fashion. This technique solves the leakage problem inherent in recent embedding methods.

## References

[1] Lev S. Dolin, Alexander G. Luchinin, and Dmitry G. Turlaev. Correction of an underwater object image distorted by surface waves. In *International Conference on Current Problems in Optics of Natural Waters*, pages 24–34, St. Petersburg, Russia, 2003.

[2] Charles Cox and Walter H. Munk. Slopes of the sea surface deduced from photographs of sun glitter. *Scripps Inst. of Oceanogr. Bull.*, 6(9):401–479, 1956.

[3] Brendan Frey and Nebojsa Jojic. Learning mixture models of images and inferring spatial transformations using the em algorithm. In *IEEE Conference on Computer Vision and Pattern Recognition*, pages 416–422, Fort Collins, June 1999.

[4] Joshua B. Tenenbaum, Vin de Silva, and John C. Langford. A global geometric framework for nonlinear dimensionality reduction. *Science*, pages 2319–2323, Dec 22 2000.

[5] Sam Roweis and Lawrence Saul. Nonlinear dimeansionality reduction by locally linear embedding. *Science*, 290(5500):2323–2326, Dec 22 2000.

[6] Bernd Fischer, Volker Roth, and Joachim M. Buhmann. Clustering with the connectivity kernel. In *Advances in Neural Information Processing Systems 16*. MIT Press, 2004.

[7] Andrew V. Goldberg. An efficient implementation of a scaling minimum-cost flow algorithm. *Journal of Algorithms*, 22:1–29, 1997.
